# Scalable imputation of genetic data with a discrete fragmentation-coagulation process

**Lloyd T. Elliott**
Gatsby Computational Neuroscience Unit
University College London
17 Queen Square
London WC1N 3AR, U.K.
elliott@gatsby.ucl.ac.uk

**Yee Whye Teh**
Department of Statistics
University of Oxford
1 South Parks Road
Oxford OX1 3TG, U.K.
y.w.teh@stats.ox.ac.uk

## Abstract

We present a Bayesian nonparametric model for genetic sequence data in which a set of genetic sequences is modelled using a Markov model of partitions. The partitions at consecutive locations in the genome are related by the splitting and merging of their clusters. Our model can be thought of as a discrete analogue of the continuous fragmentation-coagulation process [Teh et al 2011], preserving the important properties of projectivity, exchangeability and reversibility, while being more scalable. We apply this model to the problem of genotype imputation, showing improved computational efficiency while maintaining accuracies comparable to other state-of-the-art genotype imputation methods.

## 1 Introduction

The increasing availability of genetic data (for example, from the Thousand Genomes project [1]) and the importance of genetics in scientific and medical applications requires the development of scalable and accurate models for genetic sequences which are informed by genetic processes. Although standard models such as the coalescent with recombination [2] are accurate, they suffer from intractable posterior computations. To address this, various hidden Markov model (HMM) based approaches have been proposed in the literature as more scalable alternatives (e.g. [3, 4]).

Due to gene conversion and chromosomal crossover, genetic sequences exhibit a local 'mosaic'-like structure wherein sequences are composed of prototypical segments called haplotypes [5]. Locally, these prototypical segments are shared by a cluster of sequences: each sequence in the cluster is described well by a haplotype that is specific to the location on the chromosome of the cluster. An example of such a structure is shown in Figure 1. HMMs can capture this structure by having each latent state correspond to one of the haplotypes [3, 6]. Unfortunately, this leads to symmetries in the posterior distribution arising from the nonidentifiability of the state labels [7, 8]. Furthermore, current state-of-the-art HMM methods often involve costly model selection procedures in order to choose the number of latent states.

A continuous fragmentation-coagulation process (CFCP) has recently been proposed for modelling local mosaic structure in genetic sequences [9]. The CFCP is a nonparametric models defined directly on unlabelled partitions thereby avoiding both costly model selection and the label switching problem [8]. Although inference algorithms derived for the CFCP scale linearly in the number and length of the sequences [9], since the CFCP is a Markov jump process the computational overhead needed to model the arbitrary number of latent events located between two consecutive observations might preclude scalability to large datasets.

In this work, we present a novel fragmentation-coagulation process defined on a discrete grid (called the DFCP) which provides the advantages of the CFCP while being more scalable. The DFCP

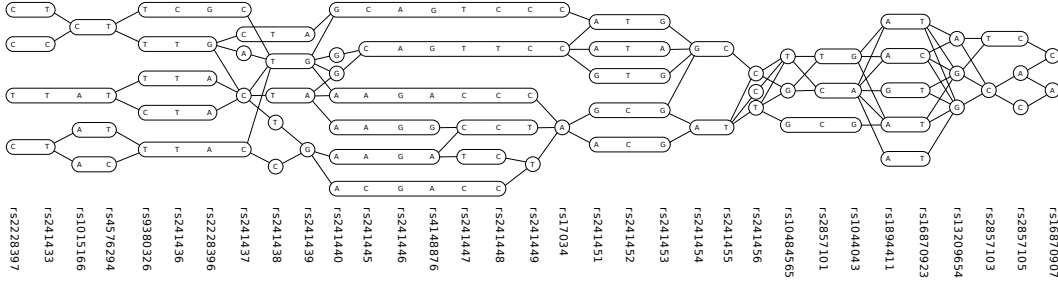

Figure 1: Haplotype structure of the CEU and YRI populations from HapMap [10] found by DFCP. Data consists of single nucleotide polymorphisms (SNPs) from TAP2 gene. Horizontal axis indicates SNP location and label. Vertical axis represents clusters from last sample of an MCMC chain converging to DFCP posterior. Letters inside clusters indicate base identity.

describes location-dependent unlabelled partitions such that at each location on the chromosome the clusters will split into multiple clusters which then merge to form the clusters at the next location. As with the CFCP, the DFCP avoids the label switching problem by defining a probability distribution directly on the space of unlabelled partitions.

The splitting and merging of clusters across the chromosome forms a mosaic structure of haplotypes. Figure 1 gives an example of the structure discovered by the DFCP. We describe the DFCP in section 2, and a forward-backward inference algorithm in section 3. Sections 4 and 5 report some experimental results showing good performance on an imputation problem, and in section 6 we conclude.

## 2 The discrete fragmentation-coagulation process

In humans, most of the bases on a chromosome are the same for all individuals in a population. Genetic variations arise through mutations such as single nucleotide polymorphisms (SNPs), which are locations in the genome where a single base was altered by a mutation at some time in the ancestry of the chromosome. At each SNP location, a particular chromosome has one of usually two possible bases (referred to as the major and minor allele). Consequently, SNP data for a chromosome can be modelled as a binary sequence, with each entry indicating which of the two bases is present at that location. In this paper we consider SNP data consisting of $n$ binary sequences $x = (x_i)_{i=1}^n$, where each sequence $x_i = (x_{it})_{t=1}^T$ is of length $T$ and corresponds to the $T$ SNPs on a segment of a chromosome in an individual. The $t$-th entry $x_{it}$ of sequence $i$ is equal to zero if individual $i$ has the major allele at location $t$ and equal to one otherwise.

We will model these sequences using a discrete fragmentation-coagulation process (DFCP) so that the sequence values at the SNP at location $t$ are described by the latent partition $\pi_t$ of the sequences. Each cluster in the partition corresponds to a haplotype. The DFCP models the sequence of partitions using a discrete Markov chain as follows: starting with $\pi_t$, we first fragment each cluster in $\pi_t$ into smaller clusters, forming a finer partition $\rho_t$. Then we coagulate the clusters in $\rho_t$ to form the clusters of $\pi_{t+1}$. In the remainder of this section, we will first give some background theory on partitions, and random fragmentation and coagulation operations and then we will describe the DFCP as a Markov chain over partitions. Finally, we will describe the likelihood model used to relate the sequence of partitions to the observed sequences.

### 2.1 Random partitions, fragmentations and coagulations

A partition of a set $S$ is a clustering of $S$ into non-overlapping non-empty subsets of $S$ whose union is all of $S$. The Chinese restaurant process (CRP) forms a canonical family of distributions on partitions. A random partition $\pi$ of a set $S$ is said to follow the law $\text{CRP}(S, \alpha, \sigma)$ if:

$$\Pr(\pi) = \frac{[\alpha + \sigma]_\sigma^{\#\pi - 1}}{[\alpha + 1]_1^{\#S - 1}} \prod_{a \in \pi} [1 - \sigma]_1^{\#a - 1} \tag{1}$$

where $[x]_d^n = (x)(x + d) \dots (x + (n - 1)d)$ is Kramp's symbol and $\alpha > -\sigma, \sigma \in [0, 1)$ are the concentration and discount parameters respectively [11]. A CRP can also be described by the following

analogy: customers (elements of $S$) enter a Chinese restaurant and choose to sit at tables (clusters in $\pi$). The first customer chooses any table. Subsequently, the $i$-th customer sits at a previously chosen table $a$ with probability proportional to $\#a - \sigma$ where $\#a$ is the number of customers already sitting there and at some unoccupied table with probability proportional to $\alpha + \sigma\#\pi$ where $\#\pi$ is the total number of tables already sat at by previous customers.

The fragmentation and coagulation operators are random operations on partitions. The fragmentation $\text{FRAG}(\pi, \alpha, \sigma)$ of a partition $\pi$ is formed by partitioning further each cluster $a$ of $\pi$ according to $\text{CRP}(a, \alpha, \sigma)$ and then taking the union of the resulting partitions, yielding a partition of $S$ that is finer than $\pi$. Conversely, the coagulation $\text{COAG}(\pi, \alpha, \sigma)$ of $\pi$ is formed by partitioning the set of clusters of $\pi$ (i.e., the set $\pi$ itself) according to $\text{CRP}(\pi, \alpha, \sigma)$ and then replacing each cluster with the union of its elements, yielding a partition that is coarser than $\pi$. The fragmentation and coagulation operators are linked through the following theorem by Pitman [12].

**Theorem 1.** *Let $S$ be a set and let $A_1, B_1, A_2, B_2$ be random partitions of $S$ such that:*
$$A_1 \sim \text{CRP}(S, \alpha\sigma_2, \sigma_1\sigma_2), \qquad B_1|A_1 \sim \text{FRAG}(A_1, -\sigma_1\sigma_2, \sigma_2),$$
$$B_2 \sim \text{CRP}(S, \alpha\sigma_2, \sigma_2), \qquad A_2|B_2 \sim \text{COAG}(B_2, \alpha, \sigma_1).$$
*Then, for all partitions $A$ and $B$ of the set $S$ such that $B$ is a refinement of $A$:*
$$\Pr(A_1 = A, B_1 = B) = \Pr(A_2 = A, B_2 = B). \tag{2}$$

## 2.2 The discrete fragmentation-coagulation process

The DFCP is parameterized by a concentration $\mu > 0$ and rates $(R_t)_{t=1}^{T-1}$ with $R_t \in [0, 1)$. Under the DFCP, the marginal distribution of the partition $\pi_t$ is $\text{CRP}(S, \mu, 0)$ and so $\mu$ controls the number of clusters that are found at each location. The rate parameter $R_t$ controls the strength of dependence between $\pi_t$ and $\pi_{t+1}$, with $R_t = 0$ implying that $\pi_t = \pi_{t+1}$ and $R_t \to 1$ implying independence.

Given $\mu$ and $(R_t)_{t=1}^{T-1}$, the DFCP on a set of sequences indexed by the set $S = \{1, \ldots, n\}$ is described by the following Markov chain. First we draw a partition $\pi_1 \sim \text{CRP}(S, \mu, 0)$. This CRP describes the clustering of $S$ at location $t = 1$. Subsequently, we draw $\rho_t|\pi_t$ from $\text{FRAG}(\pi_t, 0, R_t)$, which fragments each of the clusters in $\pi_t$ into smaller clusters in $\rho_t$, and then $\pi_{t+1}|\rho_t$ from $\text{COAG}(\rho_t, \mu/R_t, 0)$, which coagulates clusters in $\rho_t$ into larger clusters in $\pi_{t+1}$.

Each $\pi_t$ has $\text{CRP}(S, \mu, 0)$ as its invariant marginal distribution and each $\rho_t$ is marginally distributed as $\text{CRP}(S, \mu, R_t)$. This can be seen by applying Theorem 1 with the substitution $\sigma_1 = 0$, $\sigma_2 = R_t$, $\alpha = \mu/R_t$. In population genetics the CRP appears as (and was predated by) Ewen's sampling formula [13], a counting formula for the number of alleles appearing in a population, observed at a given location. Over a short segment of the chromosome where recombination rates are low, haplotypes behave like alleles and so a CRP prior on the number of haplotypes at a location is reasonable.

Further, since fragmentation and coagulation operators are defined in terms of CRPs which are projective and exchangeable, the Markov chain is projective and exchangeable in $S$ as well. Projectivity and exchangeability are desirable properties for Bayesian nonparametric models because they imply that the marginal distribution of a given data item does not depend on the total number of other data items or on the order in which the other data items are indexed. In genetics, this captures the fact that usually only a small subset of a population is observed.

Finally, the theorem also shows that conditioned on $\pi_{t+1}$, $\rho_t$ has distribution $\text{FRAG}(\pi_{t+1}, 0, R_t)$ while $\pi_t|\rho_t$ has distribution $\text{COAG}(\rho_t, \mu/R_t, 0)$ meaning that the Markov chain defining the DFCP is reversible. Chromosome replication is directional and so statistics for genetic processes along the chromosome are not reversible. But the strength of this effect on SNP data is not currently known and many genetic models such as the coalescent with recombination [14] assume reversibility for simplicity. The non-reversibility displayed by models such as fastPHASE is an artifact of their construction rather than an attempt to capture non-reversible aspects of genetic sequences.

## 2.3 Likelihood model for sequence observations

Given the sequence of partitions $(\pi_t)_{t=1}^T$, we model the observations in each cluster at each location $t$ independently. For each cluster $a \in \pi_t$ at location $t$, we adopt a discrete likelihood model in which

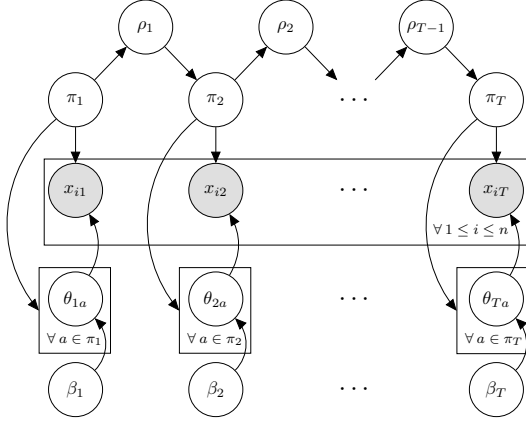

$$\pi_1 \sim \mathrm{CRP}(S, \mu, 0),$$
$$\rho_t | \pi_t \sim \mathrm{FRAG}(\pi_t, 0, R_t),$$
$$\pi_{t+1} | \rho_t \sim \mathrm{COAG}(\rho_t, \mu / R_t, 0),$$
$$\log \mu \sim \mathcal{N}(m, v),$$
$$\log R_t \sim \mathrm{Uniform}(\log R_{\min}, 0),$$
$$x_{it} | a_{it} = \theta_{ta_{it}}, \theta_{ta} | \beta_t \sim \mathrm{Bernoulli}(\beta_t),$$
$$\beta_t | \gamma_t \sim \mathrm{Beta}(\frac{\gamma_t}{2}, \frac{\gamma_t}{2}),$$
$$\log \gamma_t \sim \mathrm{Uniform}(\log \gamma_{\min}, 0). \qquad (3)$$

Figure 2: *Left:* Graphical model for the discrete fragmentation coagulation process. Hyperparameters are not shown. *Right:* Generative process for genetic sequences $x_{it}$.

the same observation is emitted for each sequence in the cluster. For each sequence $i$, let $a_{it} \in \pi_t$ be the cluster in $\pi_t$ containing $i$. Let $\theta_{ta}$ be the emission of cluster $a$ at location $t$. Since SNP data has binary labels, $\theta_{ta} \in \{0, 1\}$ is a Bernoulli random variable. Let the mean of $\theta_{ta}$ be $\beta_t$ (this is the latent allele frequency at location $t$). We assume that conditioned on the partitions and the parameters, the observations $x_{it}$ are independent, and determined by the cluster parameter $\theta_{ta}$. Thus the probability $\Pr(\theta_{ta} = 1 | \beta_t) = \beta_t$ and the probability $\Pr(x_{it} | a_{it} = a, \theta_{ta}) = \delta(x_{it} = \theta_{ta})$ where $\delta$ is an indicator function (i.e., it is one if $x_{it} = \theta_{ta}$ and zero otherwise).

We place a beta prior on $\beta_t$ with mean parameter $1/2$ and mass parameter $\gamma_t$. The mass parameters are themselves marginally independent and we place on them an uninformative log-uniform prior over a range: $p(\gamma_t) \propto \gamma_t^{-1}$, $\gamma_t \geq \gamma_{\min}$. Since this distribution is heavy tailed, the $\beta_t$ variables will have more mass near 0 and 1 than they would have if $\gamma_t$ were fixed, adding sparsity to the latent allele frequencies. This phenomenon is empirically observed in SNP data. We also place an uninformative log-uniform prior on $R_t$ over a range: $p(R_t) \propto R_t^{-1}$, $R_t \geq R_{\min}$. Note that the prior gives more mass to values of $R_t$ close to $R_{\min}$ which we set close to zero, since we expect the partitions of consecutive locations to be relatively similar so that the mosaic haplotype structure can be formed. Finally, we place a truncated log-normal prior on $\mu$ with mean $m$ and variance $v$: $\log \mu \sim \mathcal{N}(m, v), \mu > 0$. The graphical model for this generative process is shown in Figure 2.

## 2.4 Relationship with the continuous fragmentation-coagulation process

The continuous version of the fragmentation-coagulation process [9], which we refer to as the CFCP, is a partition valued Markov jump process (MJP). (The 'time' variable for this MJP is the chromosome location, viewed as a continuous variable.) The CFCP is a pure jump process and can be defined in terms of its rates for various jump events. There are two types of events in the CFCP: binary fragmentation events, in which a single cluster $a$ is split into two clusters $b$ and $c$ at a rate of $R\Gamma(\#b)\Gamma(\#c)/\Gamma(\#a)$, and binary coagulation events in which two clusters $b$ and $c$ merge to form one cluster $a$ at a rate of $R/\mu$.

As was shown in [9] the CFCP can be realised as a continuous limit of the DFCP. Consider a DFCP with concentration $\mu$ and constant rate parameter $R\varepsilon$. Then as $\varepsilon \to 0$ the probability that the coagulation and fragmentation operations at a specific time step $t$ induce no change in the partition structure $\pi_t$ approaches 1. Conversely, the probability that these operations are the binary events given above scales as $\mathcal{O}(\varepsilon)$, while all other events scale as larger powers of $\varepsilon$. If we rescale the time steps by $t \mapsto \varepsilon t$, then the expected number of binary events over a finite interval approaches $\varepsilon$ times the rates given above and the expected number of all other events goes to zero, yielding the CFCP.

In the CFCP fragmentation and coagulation events are binary: they involve either one cluster fragmenting into two new clusters, or two clusters coagulating into one new cluster. However, for the DFCP the fragmentation and coagulation operators can describe more complicated haplotype structures without introducing more latent events. For example one cluster splitting into three clusters (as happens to the second haplotype from the top of Figure 1 after the 18th SNP) can be described

by the DFCP using just one fragmentation operator. The order of the latent events introduced by the CFCP required does not matter, adding unnecessary symmetry to its posterior.

# 3   Inference with the discrete fragmentation coagulation process

We derive a Gibbs sampler for posterior simulation in the DFCP by making use of the exchangeability of the process. Each iteration of the sampler updates the trajectory of cluster assignments of one sequence $i$ through the partition structure. To arrive at the updates, we first derive the conditional distribution of the $i$-th trajectory given the others, which can be shown to be a Markov chain. Coupled with the deterministic likelihood terms, we then use a backwards-filtering/forwards-sampling algorithm to obtain a new trajectory for sequence $i$. In this section, we derive the conditional distribution of trajectory $i$ using the definition of fragmentation and coagulation and also the posterior distributions of the parameters $R_t, \mu$ which we will update using slice sampling [15].

## 3.1   Conditional probabilities for the trajectory of sequence $i$

We will refer to the projection of the partitions $\pi_t$ and $\rho_t$ onto $S - \{i\}$ by $\pi_t^{-i}$ and $\rho_t^{-i}$ respectively. Let $a_t$ (respectively $b_t$) be the cluster assignment of sequence $i$ at location $t$ in $\pi_t$ (respectively $\rho_t$). If the sequence $i$ is placed in a new cluster by itself in $\pi_t$ (i.e., it forms a singleton cluster) we will denote this by $a_t = \varnothing$ and for $\rho_t^{-i}$ we will denote the respective event by $b_t = \varnothing$. Otherwise, if the the sequence $i$ is placed in an existing cluster in $\pi_t^{-i}$ (respectively $\rho_t^{-i}$) we will denote this by $a_t \in \pi_t^{-i}$ (respectively $b_t \in \rho_t^{-i}$). Thus the state spaces of $a_t$ and $b_t$ are respectively $\pi_t^{-i} \cup \{\varnothing\}$ and $\rho_t^{-i} \cup \{\varnothing\}$.

Starting at $t = 1$, since the initial distribution is $\pi_1 \in \mathrm{CRP}(S, \mu, 0)$, the conditional cluster assignment of the sequence $i$ in $\pi_1$ is given by the CRP probabilities from (1):

$$\Pr(a_t = a | \pi_1^{-i}) = \begin{cases} \#a/(n - 1 + \mu) & \text{if } a \in \pi_t^{-i}, \\ \mu/(n - 1 + \mu) & \text{if } a = \varnothing. \end{cases} \tag{4}$$

To find the conditional distribution of $b_t$ given $a_t$, we use the definition of the fragmentation operation as independent CRP partitions of each cluster in $\pi_t$. If $a_t = \varnothing$, then the sequence $i$ is in a cluster by itself in $\pi_t$ and so it will remain in a cluster by itself after fragmenting. Thus, $b_t = \varnothing$ with probability 1. If $a_t = a \in \pi_t^{-i}$ then $b_t$ must be one of the clusters in $\rho_t$ into which $a$ fragments. This can be a singleton cluster, in which case $b_t = \varnothing$, or it can be one of the clusters in $\rho_t^{-i}$. We will refer to this set of clusters in $\rho_t^{-i}$ by $\mathcal{F}_t(a)$. Since $a$ is fragmented according to $\mathrm{CRP}(a, 0, R)$, when the $i$-th sequence is added to this CRP it is placed in a cluster $b \in \mathcal{F}_t(a)$ with probability proportional to $(\#b - R)$ and it is placed in a singleton cluster with probability proportional to $R\#\mathcal{F}_t(a)$. Normalizing these probabilities yields the following joint distribution:

$$\Pr(b_t = b | a_t = a, \pi_t^{-i}, \rho_t^{-i}) = \begin{cases} (\#b - R_t)/\#a & \text{if } a \in \pi_t^{-i}, b \in \mathcal{F}_t(a), \\ R_t \#\mathcal{F}_t(a)/\#a & \text{if } a \in \pi_t^{-i}, b = \varnothing, \\ 1 & \text{if } a = b = \varnothing, \\ 0 & \text{otherwise.} \end{cases} \tag{5}$$

Similarly, to find the conditional distribution of $a_{t+1}$ given $b_t = b$ we use the definition of the coagulation operation. If $b \neq \varnothing$, then the sequence $i$ was not in a singleton cluster in $\rho_t^{-i}$ and so it must follow the rest of the sequences in $b$ to the unique $a \in \pi_{t+1}^{-i}$ such that $b \subseteq a$ (i.e., $b$ coagulates with other clusters to form $a$). We will refer to the set of clusters in $\rho_t^{-i}$ that coagulate to form $a$ by $\mathcal{C}_t(a)$. If $b = \varnothing$ then the sequence $i$ is in a singleton cluster in $\rho_t^{-i}$ and so we can imagine it being the last customer added to the coagulating $\mathrm{CRP}(\rho_t, \mu/R_t, 0)$ of the clusters of $\rho_t$. Hence the probability that sequence $i$ is placed in a cluster $a \in \pi_{t+1}^{-i}$ is proportional to $\#\mathcal{C}_t(a)$ while the probability that it forms a cluster by itself in $\pi_{t+1}^{-i}$ is proportional to $\mu/R_t$. This yields the following joint probability:

$$\Pr(a_{t+1} = a | b_t = b, \pi_{t+1}^{-i}, \rho_t^{-i}) = \begin{cases} 1 & \text{if } a \in \pi_{t+1}^{-i}, b \in \mathcal{C}_t(a), \\ R_t \#\mathcal{C}_t(a)/(\mu + R_t \#\rho_t^{-i}) & \text{if } a \in \pi_{t+1}^{-i}, b = \varnothing, \\ \mu/(\mu + R_t \#\rho_t^{-i}) & \text{if } a = b = \varnothing, \\ 0 & \text{otherwise.} \end{cases} \tag{6}$$

## 3.2 Message passing and sampling for the sequences of the DFCP

Once the conditional probabilities are defined, it is straightforward to derive messages that allow us to conduct backwards-filtering/forwards-sampling to resample the trajectory of sequence $i$ in the DFCP. This provides an exact Gibbs update for the trajectory of that sequence conditioned on the trajectories of all the other sequences and the data. The messages we will define are the conditional distribution of all the data seen after a given location in the sequence conditioned on the cluster assignment of sequence $i$ at that location. The messages are defined as follows:

$$m_{\mathcal{C}}^t(a) = \Pr(x_{i,(t+1):T}|a_t = a, \pi_{t:T}^{-i}, \rho_{t:(T-1)}^{-i}). \tag{7}$$

$$m_{\mathcal{F}}^t(b) = \Pr(x_{i,(t+1):T}|b_t = b, \pi_{t:T}^{-i}, \rho_{t:(T-1)}^{-i}). \tag{8}$$

We define the last messages to be $m_{\mathcal{C}}^T(a) = 1$. These messages are computed as follows:

$$m_{\mathcal{F}}^t(b) = \sum_{a \in \pi_{t+1}^{-i} \cup \{\varnothing\}} m_{\mathcal{C}}^{t+1}(a) \underbrace{\delta(x_{i,(t+1)} = \theta_{(t+1),a})}_{\text{Likelihood.}} \underbrace{\Pr(a_{t+1} = a|b_t = b, \pi_{t+1}^{-i}, \rho_t^{-i})}_{\text{Coagulation probabilities from (6).}}. \tag{9}$$

$$m_{\mathcal{C}}^t(a) = \sum_{b \in \rho_t^{-i} \cup \{\varnothing\}} m_{\mathcal{F}}^t(b) \underbrace{\Pr(b_t = b|a_t = a, \pi_t^{-i}, \rho_t^{-i})}_{\text{Fragmentation probabilities from (5).}}. \tag{10}$$

As the fragmentation and coagulation conditional probabilities are only supported for clusters $a, b$ such that $b \subseteq a$, these sums can be expanded so that only non-zero terms are summed over. For simplicity we do not provide these expanded forms here. Given these computations it is easy to define backwards messages using the reversibility of the process. The backwards messages can be used to compute marginal probabilities of the observation as in the forward-backward algorithm.

To sample from the posterior distribution of the trajectory for sequence $i$ conditioned on the other trajectories and the data, we use the Markov property for the chain $a_1, b_1, \ldots, b_{T-1}, a_T$ and the definition of the messages. Starting at location 1, we have:

$$\Pr(a_1 = a|x_i, \pi_{1:T}^{-i}, \rho_{1:(T-1)}^{-i})$$
$$\propto \Pr(a_1 = a|\pi_1^{-i}) \Pr(x_{i1}|a_1 = a) \Pr(x_{i,2:T}|a_1 = a, \pi_{1:T}^{-i}, \rho_{1:(T-1)}^{-i}),$$
$$= \underbrace{\Pr(a_1 = a|\pi_1^{-i})}_{\text{CRP probabilities (1).}} \underbrace{\delta(x_1 = \theta_{1a})}_{\text{Likelihood.}} m_{\mathcal{C}}^1(a). \tag{11}$$

For subsequent $b_t$ and $a_{t+1}$ for locations $t = 1, \ldots, T-1$,

$$\Pr(b_t = b|a_t = a, x_i, \pi_{1:T}^{-i}, \rho_{1:(T-1)}^{-i})$$
$$\propto \Pr(b_t = b|a_t = a, \pi_t^{-i}, \rho_t^{-i}) \Pr(x_{i,(t+1):T}|b_t = b, \pi_{t:T}^{-i}, \rho_{t:(T-1)}^{-i}),$$
$$= \underbrace{\Pr(b_t = b|a_t = a, \pi_t^{-i}, \rho_t^{-i})}_{\text{Fragmentation probabilities from (5).}} m_{\mathcal{F}}^t(b). \tag{12}$$

$$\Pr(a_t = a|b_{t-1} = b, x_i, \pi_{1:T}^{-i}, \rho_{1:(T-1)}^{-i})$$
$$\propto \Pr(a_t = a|b_{t-1} = b, \pi_t^{-i}, \rho_{t-1}^{-i}) \Pr(x_{it}|a_t = a) \Pr(x_{i,(t+1):T}|a_t = a, \pi_{t:T}^{-i}, \rho_{t:(T-1)}^{-i}),$$
$$= \underbrace{\Pr(a_t = a|b_{t-1} = b, \pi_t^{-i}, \rho_{t-1}^{-i})}_{\text{Coagulation probability from (6).}} \underbrace{\delta(x_{it} = \theta_{ta})}_{\text{Likelihood.}} m_{\mathcal{C}}^t(a). \tag{13}$$

The complexity of this update is $\mathcal{O}(KT)$ where $K$ is the expected number of clusters in the posterior. This complexity class is the same as for the continuous fragmentation-coagulation process and other related HMM methods such as fastPHASE. But there is no exact Gibbs update for the trajectories in the CFCP. Instead the CFCP sampler relies on uniformization [16] which has slower mixing times than exact Gibbs and so the update for the DFCP is, theoretically, more efficient.

### 3.3 Parameter updates

We use slice sampling [15] to update the $\mu$ and $R_t$ parameters conditioned on the partition structure. Using Bayes' rule, the definition (3) of the DFCP, and the identity $[a]_b^n = b^n \Gamma(a/b + n)/\Gamma(a/b)$,

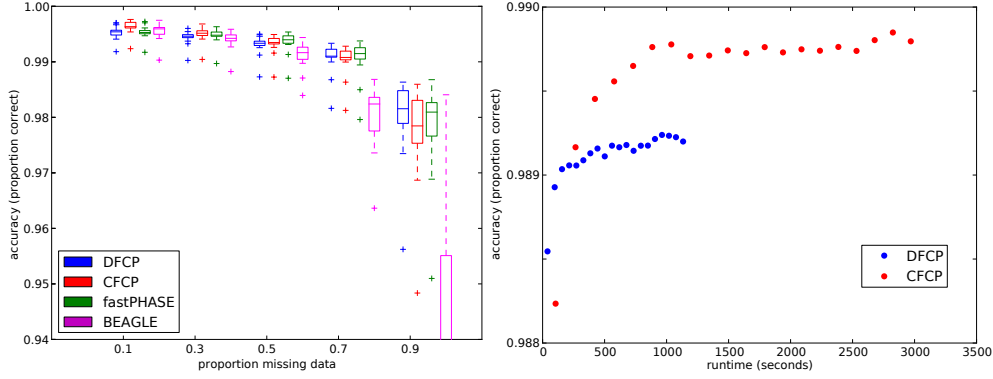

Figure 3: Allele imputation for X chromosomes from the Thousand Genomes project. *Left:* Accuracy for prediction of held out alleles for continuous (CFCP) and discrete (DFCP) versions of fragmentation-coagulation process and for popular methods BEAGLE and fastPHASE. 90% missing data condition truncates BEAGLE accuracies to emphasize other conditions. *Right:* Runtime versus accuracy for 500 MCMC iterations for DFCP and CFCP in 50% missing data condition. Points are averaged over 20 datasets and 25 consecutive samples.

the posterior probabilities of $\mu$ and $R_t$ given the partitions $\pi_{1:T}$ and $\rho_{1:(T-1)}$ are as follows:

$$\Pr(\mu|\pi,\rho) \propto \Pr(\mu)\Pr(\pi_1|\mu,R_1)\Pr(\rho_1|\pi_1,\mu,R_1)\cdots\Pr(\pi_T|\rho_{T-1},\mu,R_{T-1}),$$

$$\propto \Pr(\mu)\frac{\Gamma(\mu)}{\Gamma(\mu+n)}\mu^{-T+\sum_{t=1}^T \#\pi_t}\prod_{t=1}^{T-1}\frac{\Gamma(\mu/R_t)}{\Gamma(\mu/R_t+\#\rho_t)}. \quad (14)$$

$$\Pr(R_t|\pi,\rho,\mu) \propto \Pr(R_t)\Pr(\rho_t|\pi_t,\mu,R_t)\Pr(\pi_{t+1}|\rho_t,\mu,R_t),$$

$$\propto \Pr(R_t)R_t^{\#\rho_t-\#\pi_t-\#\pi_{t+1}+1}\frac{\Gamma(\mu/R_t)\Gamma(1-R_t)^{-\#\rho_t}}{\Gamma(\#\rho_t+\mu/R_t)}\prod_{b\in\rho_t}\Gamma(\#b-R_t). \quad (15)$$

## 4  Experiments

To examine the accuracy and scalability of the DFCP we conducted an allele imputation experiment on SNP data from the Thousand Genomes project[1]. We also compared the runtime of the samplers for the DFCP and CFCP on data simulated from the coalescent with recombination model [14]. In this section, we describe the setup of these experiments and in section 5 we present the results.

For the allele imputation experiment, we considered SNPs from 524 male X chromosomes. We chose 20 intervals randomly, each containing 500 consecutive SNPs. In five conditions we held out nested sets of between 10% and 90% of the alleles uniformly over all pairs of sites and individuals, and used fastPHASE [3], BEAGLE [17], CFCP [9] and the DFCP to predict the held out alleles.

We used the most recent versions of BEAGLE and fastPHASE software available to us. We implemented the DFCP with many of the same libraries and programming techniques as the CFCP and both versions were optimized. In each missing data condition, the CFCP and DFCP were run with five random restarts and 46 MCMC iterations per restart (26 of which were discarded for burnin and thinning). The accuracies for the DFCP and CFCP were computed by thresholding the empirical marginal probabilities of the held out alleles at 0.5. The priors on the hyper parameters and the likelihood specification of the two models were matched and the samplers were initialized using a sequential Monte Carlo method based on the trajectory updates.

The posterior distributions of the concentration parameter $\mu$ for the two methods are different. In order to match the expected number of clusters in the posterior, we also conducted allele imputation in the 50% missing data condition with $\mu$ fixed at 10.0 for both models. We simulated 500 MCMC iterations with no random restarts. We then computed the accuracy of the samples by predicting held out alleles based on the cluster assignments of the sample.

In a second experiment we simulated datasets from the coalescent with recombination model consisting of between 10,000 and 50,000 sequences using the software ms [14]. We conducted posterior MCMC simulation in both models and compared the computation time required per iteration.

# 5 Results

The accuracy of the DFCP in the allele imputation experiment was comparable to that of the CFCP and fastPHASE in all missing data conditions (Figure 3, left). For the 70% and 90% missing data conditions, BEAGLE performed poorly (its median accuracy for this condition was 93.90% and mean at chance accuracy for all conditions was 93.44%). In Figure 3(right) we compare the accuracy and runtime for the 50% missing data condition. This figure shows that the runtime required for each iteration is lower for the DFCP and the sequential Monte Carlo initialization is better (i.e., closer to a posterior mode) for the DFCP. No difference in mixing time is suggested by the figure. As an aside, we estimated the Shannon entropy in these samples and found that the DFCP had slightly more entropy per sample than the CFCP (the difference was small but statistically significant). This could indicate that the DFCP has better mixing.

For the second experiment, we plot the runtime per iteration of both models against the number of sequences in the simulated dataset (Figure 4). The DFCP was around 2.5 times faster than the CFCP for the condition with 50,000 sequences. In both models, most of the computation time was spent calculating the messages in the backwards-filtering step. The CFCP has an arbitrary number of latent events between consecutive observations and it is likely that the runtime improvement shown by the DFCP is due to its reduced number of required message calculations.

# 6 Discussion

In this paper we have presented a discrete fragmentation-coagulation process. The DFCP is a partition-valued Markov chain, where partitions change along the chromosome by a fragmentation operation followed by a coagulation operation. The DFCP is designed to model the mosaic haplotype structure observed in genetic sequences. We applied the DFCP to an allele prediction task on data from the Thousand Genomes Project yielding accuracies comparable to state-of-the-art methods and runtimes that were lower than the runtimes of the continuous fragmentation-coagulation process [9].

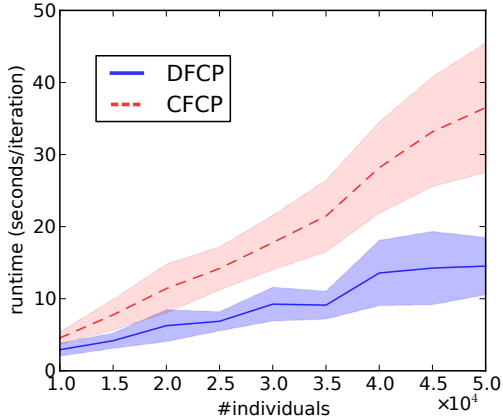

Figure 4: Runtimes per iteration per sequence of DFCP and CFCP on simulated datasets consisting of large numbers of sequences. Lines indicate mean. Shaded region indicates standard deviation.

The DFCP and CFCP induce different joint distributions on the partitions at adjacent locations. The CFCP is a Markov jump process with an arbitrary number of latent binary events wherein a single cluster is split into two clusters, or two clusters are merged into one. The DFCP however can model any partition structure with one pair of fragmentation and coagulation operations. Exact Gibbs updates for the partitions are possible in the DFCP whereas sampling in the CFCP uses uniformization [16] which, although fast in practice, has in theory slower mixing than exact Gibbs.

In future work we will explore better calling and calibration methods to improve imputation accuracies. Another avenue of future research is to understand how other genetic processes can be incorporated into the fragmentation-coagulation framework, including population admixture and gene conversion. Although haplotype structure is a local property, the Markov assumption does not hold in real genetic data. This could be reflected through hierarchical FCP models or adaptation of other dependent nonparametric models such as the spatially normalized Gamma process [18].

### Acknowledgements

We thank the Gatsby Charitable Foundation for funding. We also thank Andriy Mnih, Vinayak Rao and Anna Goldenberg for helpful discussion and the anonymous reviewers for their suggestions.

## Footnotes

[1]March 2012 v3 release of the Thousand Genomes Project.

# References

[1] The 1000 Genomes Project Consortium. A map of human genome variation from population-scale sequencing. *Nature*, 467:1061–1073, 2010.

[2] R. R. Hudson. Properties of a neutral allele model with intragenic recombination. *Theoretical Population Biology*, 23(2):183 – 201, 1983.

[3] P. Scheet and M. Stephens. A fast and flexible statistical model for large-scale population genotype data: Applications to inferring missing genotypes and haplotypic phase. *The American Journal of Human Genetics*, 78(4):629 – 644, 2006.

[4] J. Marchini, B. Howie, S. Myers, G. McVean, and P. Donnelly. A new multipoint method for genome-wide association studies by imputation of genotypes. *Nature Genetics*, 39(7):906–913, 2007.

[5] M. J. Daly, J. D. Rioux, S. F. Schaffner, T. J. Hudson, and R. S. Lander. High-resolution haplotype structure in the human genome. *Nature Genetics*, 29:229–232, 2001.

[6] J. Marchini, D. Cutler, N. Patterson, M. Stephens, E. Eskin, E. Halperin, S. Lin, Z.S. Qin, H.M. Munro, G.R. Abecasis, P. Donnelly, and the International HapMap Consortium. A comparison of phasing algorithms for trios and unrelated individuals. *The American Journal of Human Genetics*, 78(3):437 – 450, 2006.

[7] M. Stephens. Dealing with label switching in mixture models. *Journal of the Royal Statistical Society: Series B (Statistical Methodology)*, 62(4):795–809, 2000.

[8] A. Jasra, C. C. Holmes, and D. A. Stephens. Markov chain Monte Carlo methods and the label switching problem in Bayesian mixture modeling. *Statistical Science*, 20(1):50–67, 2005.

[9] Y. W. Teh, C. Blundell, and L. T. Elliott. Modelling genetic variations using fragmentation-coagulation processes. In *Advances in neural information processing systems*, 2011.

[10] The International HapMap Consortium. The international HapMap project. *Nature*, 426:789–796, 2003.

[11] J. Pitman. *Combinatorial stochastic processes*. Springer-Verlag, 2006.

[12] J. Pitman. Coalescents with multiple collisions. *Annals of Probability*, 27:1870–1902, 1999.

[13] W. J. Ewens. The sampling theory of selectively neutral alleles. *Theoretical Population Biology*, 3:87–112, 1972.

[14] R. R. Hudson. Generating samples under a Wright-Fisher neutral model of genetic variation. *Bioinfomatics*, 18:337–338, 2002.

[15] R. M. Neal. Slice sampling. *Annals of Statistics*, 31:705–767, 2003.

[16] V. Rao and Y. W. Teh. Fast MCMC sampling for Markov jump processes and continuous time Bayesian networks. In *Proceedings of the International Conference on Uncertainty in Artificial Intelligence*, 2011.

[17] B. L. Browning and S. R. Browning. A unified approach to genotype imputation and haplotype-phase inference for large data sets of trios and unrelated individuals. *American Journal of Human Genetics*, 84:210–223, 2009.

[18] V. Rao and Y. W. Teh. Spatial normalized gamma processes. In *Advances in Neural Information Processing Systems*, volume 22, pages 1554–1562, 2009.

